# AN APPLICATION OF THE PRINCIPLE OF MAXIMUM INFORMATION PRESERVATION TO LINEAR SYSTEMS

Ralph Linsker
IBM T. J. Watson Research Center, Yorktown Heights, NY 10598

## ABSTRACT

This paper addresses the problem of determining the weights for a set of linear filters (model "cells") so as to maximize the ensemble-averaged information that the cells' output values jointly convey about their input values, given the statistical properties of the ensemble of input vectors. The quantity that is maximized is the Shannon information rate, or equivalently the average mutual information between input and output. Several models for the role of processing noise are analyzed, and the biological motivation for considering them is described. For simple models in which nearby input signal values (in space or time) are correlated, the cells resulting from this optimization process include center-surround cells and cells sensitive to temporal variations in input signal.

## INTRODUCTION

I have previously proposed [Linsker, 1987, 1988] a principle of "maximum information preservation," also called the "infomax" principle, that may account for certain aspects of the organization of a layered perceptual network. The principle applies to a layer L of cells (which may be the input layer or an intermediate layer of the network) that provides input to a next layer M. The mapping of the input signal vector L onto an output signal vector M, $f:L \rightarrow M$, is characterized by a conditional probability density function ("pdf") $P(M|L)$. The set S of allowed mappings $f$ is specified. The input pdf $P_L(L)$ is also given. (In the cases considered here, there is no feedback from M to L.) The infomax principle states that a mapping $f$ should be chosen for which the Shannon information rate [Shannon, 1949]

$$R(f) \equiv \int dL \, P_L(L) \int dM \, P(M|L) \, \log[P(M|L)/P_M(M)] \qquad (1)$$

is a maximum (over all $f$ in the set S). Here $P_M(M) \equiv \int dL P_L(L)P(M|L)$ is the pdf of the output signal vector M. $R$ is identical to the average mutual information between L and M.

To understand better how the infomax principle may be applied to biological systems and complex synthetic networks, it is useful to solve the infomax optimization problem explicitly for simpler systems whose properties are nonetheless biologically motivated. This paper therefore deals with the practical computation of infomax solutions for cases in which the mappings $f$ are constrained to be linear.

## INFOMAX SOLUTIONS FOR A SET OF LINEAR FILTERS

We consider the case of linear model "neurons" with multivariate Gaussian input and additive Gaussian noise. There are $N$ input (L) cells and $N'$ output (M) cells. The input column vector $L \equiv (L_1, L_2, \ldots, L_N)^T$ is randomly selected from an $N$-dimensional Gaussian distribution having mean zero. That is,

$$P_L(L) = (2\pi)^{-N/2}(\text{Det } Q^L)^{-1/2} \exp[-(1/2)L^T(Q^L)^{-1}L] \tag{2}$$

where $Q^L$ is the covariance matrix of the input activities, $Q_{ij}^L \equiv \int dL\, P_L(L)L_iL_j$. (Superscript $T$ denotes the matrix transpose.)

To specify the set S of allowed mappings $f:L \rightarrow M$, we define a processing model that includes a description of (i) how noise enters during processing, (ii) the independent variables over which we are to maximize $R$, and (iii) any constraints on their values. Figure 1 shows several such models. We shall analyze the simplest, then explain the motivation for the more complex models and analyze them in turn.

**Model A -- Additive noise of constant variance**

In Model A of Fig. 1 the output signal value of the $n$th M cell is:

$$M_n = \Sigma_i C_{ni}L_i + \nu_n. \tag{3}$$

The noise components $\nu_n$ are independently and identically distributed ("i.i.d.") random variables drawn from a Gaussian distribution having a mean of zero and variance $B$.

Each mapping $f:L \rightarrow M$ is characterized by the values of the $\{C_{ni}\}$ and the noise parameter $B$. The elements of the covariance matrix of the output activities are (using Eqn. 3)

$$Q_{nm}^M(f) \equiv \int dM\, P_M(M)\, M_nM_m = B\delta_{nm} + \Sigma_{i,j}C_{ni}Q_{ij}^L C_{mj}, \tag{4}$$

where $\delta_{nm} \equiv 1$ if $n = m$ and 0 otherwise.

Evaluating Eqn. 1 for this processing model gives the information rate:

$$R(f) = (1/2)\, \ln \text{Det } W(f) \tag{5}$$

where $W_{nm} \equiv Q_{nm}^M/B$. ($R$ is the difference of two entropy terms. See [Shannon, 1949], p.57, for the entropy of a Gaussian distribution.)

If the components $C_{ni}$ of the $C$ matrix are allowed to be arbitrarily large, then the information rate can be made arbitrarily large, and the effects of noise become arbitrarily small. One way to limit $C$ is to impose a "resource constraint" on each M cell. An example of such a constraint is $\Sigma_i C_{ni}^2 = 1$ for all $n$. One can then attempt directly, using numerical methods, to maximize Eqn. 5 over all allowed $C$ for given $B$. However, when some additional conditions (below) are satisfied, further analytical progress can be made.

Suppose the $N$ L-cells are uniformly spaced along the line interval $[0,1]$ with periodic boundary conditions, so that cell $N$ is next to cell 1. [The analysis can be extended to a two- (or higher-) dimensional array in a straightforward manner.] Suppose also that (for given $N$) the covariance $Q_{ij}^L$ of the input values at cells $i$ and $j$ is a function $Q^L(s_{ij})$ only of the displacement $s_{ij}$ from $i$ to $j$. (We deal with the periodicity by defining $s_{ab} \equiv b - a - \gamma_{ab}N$ and choosing the integer $\gamma_{ab}$ such that $-N/2 \leq s_{ab} < N/2$.) Then $Q^L$ is a Toeplitz matrix, and its eigenvalues $\{\lambda_k\}$ are the components of the discrete Fourier transform ("F.T.") of $Q^L(s)$:

$$\lambda_k = \Sigma_s Q^L(s) \exp( -2\pi\sqrt{-1} \, ks/N), \quad ( -N/2) \leq k < N/2. \tag{6}$$

We now impose two more conditions: (1) $N' = N$. This simplifies the resulting expressions, but is otherwise inessential, as we shall discuss. (2) We constrain each M cell to have the same arrangement of $C$-values relative to the M cell's position. That is, $C_{ni}$ is to be a function $C(s_{ni})$ only of the displacement $s_{ni}$ from $n$ to $i$. This constraint substantially reduces the computational demands. We would not expect

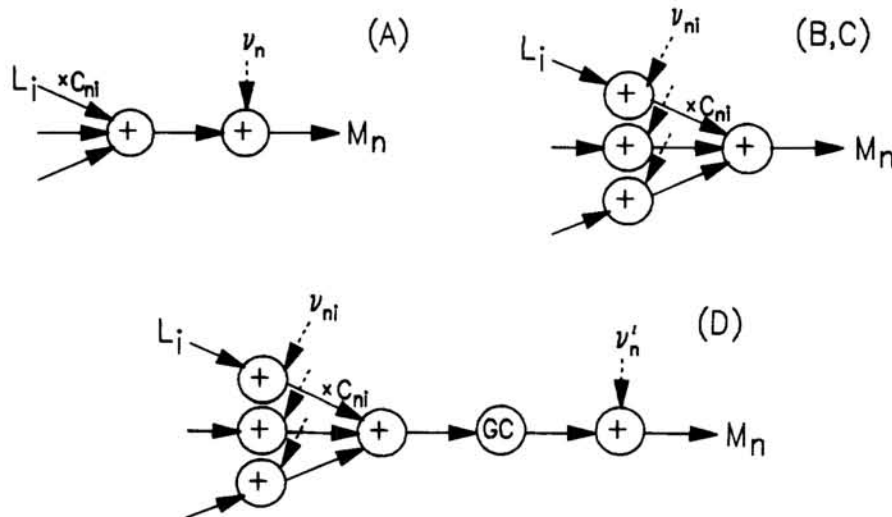

Figure 1.    Four processing models (A)-(D):   Each diagram shows a single M cell (indexed by $n$) having output activity $M_n$. Inputs $\{L_i\}$ may be common to many M cells. All noise contributions (dotted lines) are uncorrelated with one another and with $\{L_i\}$. GC = gain control (see text).

it to hold in general in a biologically realistic model -- since different M cells should be allowed to develop different arrangements of weights -- although even then it could be used as an Ansatz to provide a lower bound on $R$. The section, "Temporally-correlated input patterns," deals with a situation in which it is biologically plausible to impose this constraint.

Under these conditions, $(Q_{nm}^M)$ is also a Toeplitz matrix. Its eigenvalues are the components of the F.T. of $Q^M(s_{nm})$. For $N' = N$ these eigenvalues are $(B + \lambda_k z_k)$, where $z_k \equiv |c_k|^2$ and $c_k \equiv \Sigma_s C(s) \exp(-2\pi\sqrt{-1} ks/N)$ is the F.T. of $C(s)$. [This expression for the eigenvalues is obtained by rewriting Eqn. 4 as: $Q^M(s_{nm}) = B\delta_{n-m,0} + \Sigma_{i,j}C(s_{ni})Q^L(s_{ij})C(s_{mj})$ , and taking the F.T. of both sides.] Therefore

$$R = (1/2)\Sigma_k \ln[1 + \lambda_k z_k/B]. \qquad (7)$$

We want to maximize $R$ subject to $\Sigma_s C(s)^2 = 1$, which is equivalent to $\Sigma z_k = N$. Using the Lagrange multiplier method, we maximize $A \equiv R + \mu(\Sigma z_k - N)$ over all nonnegative $\{z_k\}$ . Solving for $\partial A/\partial z_k = 0$ and requiring $z_k \geq 0$ for all $k$ gives the solution:

$$z_k = \max[(-1/2\mu) - (B/\lambda_k), 0], \qquad (8)$$

where (given $B$) $\mu$ is chosen such that $\Sigma z_k = N$.

Note that while the optimal $\{z_k\}$ are uniquely determined, the phases of the $\{c_k\}$ are completely arbitrary [except that since the $\{C(s)\}$ are real, we must have $c_k^* = c_{-k}$ for all $k$]. The $\{C(s)\}$ values are therefore not uniquely determined. Fig. 2a shows two of the solutions for an example in which $Q^L(s) = \exp[-(s/s_0)^2]$ with $s_0 = 6$, $N = N' = 64$, and $B \doteq 1$. Both solutions have $z_{0,\pm1,...,\pm6}$=5.417, 5.409, 5.378, 5.306, 5.134, 4.689, 3.376, and all other $z_k \equiv 0$. Setting all $c_k$ phases to zero yields the solid curve; a particular random choice of phases yields the dotted curve. We shall later see that imposing locality conditions on the $\{C(s)\}$ (e.g., penalizing nonzero $C(s)$ for large $|s|$) can remove the phase ambiguity.

Our solution (Eqn. 8) can be described in terms of a so-called "water-filling" analogy: If one plots $B/\lambda_k$ versus $k$, then $z_k$ is the depth of "water" at $k$ when one "pours" into the "vessel" defined by the $B/\lambda_k$ curve a total quantity of "water" that corresponds to $\Sigma z_k = N$ and brings the "water level" to $(-1/2\mu)$.

Let us contrast this problem with two other problems to which the "water-filling" analogy has been applied in the information-theory literature. In our notation, they are:

1.    Given a transfer function $\{C(s)\}$ and the noise variance $B$, how should a given total input signal power $\Sigma\lambda_k$ be apportioned among the various wavenumbers $k$ so as to maximize the information rate $R$ [Gallager, 1968]? Our problem is complementary to this: we fix the input signal properties and seek an optimal transfer function subject to constraints.

2.    Rate-distortion (R-D) calculation [Berger, 1971]: Given a distortion measure (that defines a "distance" between the actual input signal and an estimate of it that can be reconstructed from the channel's output), and the input power spectrum $\{\lambda_k\}$, what choice of $\{z_k\}$ minimizes the average distortion for given information rate (or minimizes the required rate for given distortion)? In the R-D problem there is a process of reconstruction, and a given measure for assessing the "goodness" of reconstruction. In contrast, in our network there is no reconstruction of the input signal, and no criterion of the "goodness" of such a hypothetical reconstruction is provided.

Note also that infomax optimization is not the same as computing which channel (that is, which mapping $f:L \to M$) selected from an allowed set has the maximum information-theoretic capacity. In that problem, one is free to encode the inputs before transmission so as to make optimal use of (i.e., "achieve the capacity of") the channel. In our case, there is no such pre-encoding; the input ensemble is prescribed (by the environment or by the output of an earlier processing stage) and we need to maximize the channel rate for that ensemble.

The simplifying condition that $N = N'$ (above) is unnecessarily restrictive. Eqn. 7 can be easily generalized to the case in which $N$ is a multiple of $N'$ and the $N'$ M cells are uniformly spaced on the unit interval. Moreover, in the limit that $1/N'$ is much smaller than the correlation length scale of $Q^L$, it can be shown that $R$ is unchanged when we simultaneously increase $N'$ and $B$ by the same factor. (For example, two adjacent M cells each having noise variance $2B$ jointly convey the same information

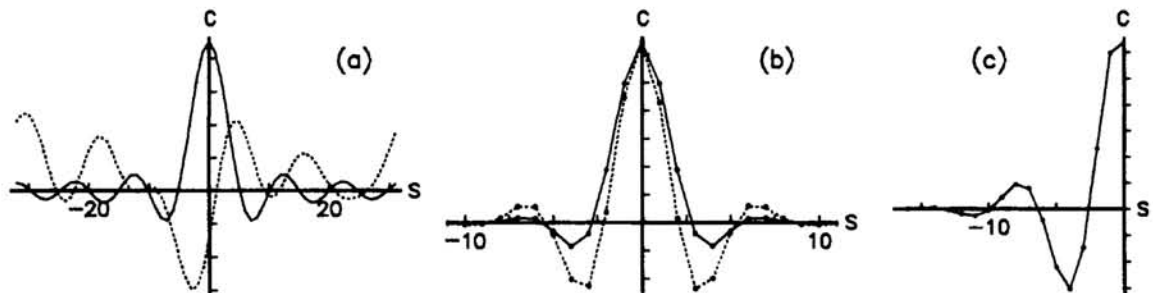

Figure 2.    Example infomax solutions $C(s)$ for locally-correlated inputs: (a) Model A; region of nonnegligible $C(s)$ extends over all $s$; phase ambiguity in $c_k$ yields nonunique $C(s)$ solutions, two of which are shown. See text for details. (b) Models C (solid curve) and D (dotted curve) with Gaussian $g(s)^{-1}$ favoring short connections; shows center-surround receptive fields, more pronounced in Model D. (c) "Temporal receptive field" using Model D for temporally correlated scalar input to a single M cell; $C(s)$ is the weight applied to the input signal that occurred $s$ time steps ago. Spacing between ordinate marks is 0.1; $\Sigma C(s)^2 = 1$ in each case.

about L as one M cell having noise variance $B$.) For biological applications we are mainly interested in cases in which there are many L cells [so that $C(s)$ can be treated as a function of a continuous variable] and many M cells (so that the effect of the noise process is described by the single parameter $B/N'$).

The analysis so far shows two limitations of Model A. First, the constraint $\Sigma_i C_{ni}^2 = 1$ is quite arbitrary. (It certainly does not appear to be a biologically natural constraint to impose!) Second, for biological applications we are interested in predicting the favored values of $\{C(s)\}$, but the phase ambiguity prevents this. In the next section we show that a modified noise model leads naturally, without arbitrary constraints on $\Sigma_i C_{ni}^2$, to the same results derived above. We then turn to a model that favors local connections over long-range ones, and that resolves the phase ambiguity issue.

**Model B -- Independent noise on each input line**

In Model B of Fig. 1 each input $L_i$ to the $n$th M cell is corrupted by i.i.d. Gaussian noise $\nu_{ni}$ of mean zero and variance $B$. The output is

$$M_n = \Sigma_i C_{ni}(L_i + \nu_{ni}). \tag{9}$$

Since each $\nu_{ni}$ is independent of all other noise terms (and of the inputs $\{L_i\}$), we find

$$Q_{nm}^M = \Sigma_{i,j} C_{ni} Q_{ij}^L C_{mj} + B\delta_{nm} \Sigma_i C_{ni}^2. \tag{10}$$

We may rewrite the last term as $B\delta_{nm} (\Sigma_i C_{ni}^2)^{1/2} (\Sigma_j C_{mj}^2)^{1/2}$. The information rate is then $R = (1/2) \ln \mathrm{Det} W$ where

$$W_{nm} = \delta_{nm} + (\Sigma_{i,j} C_{ni} Q_{ij}^L C_{mj})/[B (\Sigma_i C_{ni}^2)^{1/2} (\Sigma_j C_{mj}^2)^{1/2}]. \tag{11}$$

Define $C'_{ni} \equiv C_{ni}(\Sigma_k C_{nk}^2)^{-1/2}$ ; then $W_{nm} = \delta_{nm} + (\Sigma_{i,j} C'_{ni} Q_{ij}^L C'_{mj})/B$. Note that this is identical (except for the replacement $C \rightarrow C'$) to the expression following Eqn. (5), in which $Q^M$ was given by Eqn. (4). By definition, the $\{C'_{ni}\}$ satisfy $\Sigma_i C'^2_{ni} = 1$ for all $n$. Therefore, the problem of maximizing $R$ for this model (with no constraints on $\Sigma_i C_{ni}^2$) is identical to the problem we solved in the previous section.

**Model C -- Favoring of local connections**

Since the arborizations of biological cells tend to be spatially localized in many cases, we are led to consider constraints or cost terms that favor localization. There are various ways to implement this. Here we present a way of modifying the noise process so that the infomax principle itself favors localized solutions, without requiring additional terms unrelated to information transmission.

Model C of Fig. 1 is the same as Model B, except that now the longer connections are "noisier" than the shorter ones. That is, the variance of $\nu_{ni}$ is $<\nu_{ni}^2> = B_0 g(s_{ni})$ where $g(s)$ increases with $|s|$. [Equivalently, one could attenuate the signal on the $(i \rightarrow n)$ line by $g(s_{ni})^{1/2}$ and have the same noise variance $B_0$ on all lines.]

This change causes the last term of Eqn. 10 to be replaced by $B_0\delta_{nm}\Sigma_i g(s_{ni})C_{ni}^2$. Under the conditions discussed earlier (Toeplitz $Q^L$ and $Q^M$, and $N = N'$), we derive

$$R = (1/2)\Sigma_k \ln\{1 + \lambda_k |c_k|^2/[B_0\Sigma_s g(s)C(s)^2]\}. \tag{12}$$

Recall that the $\{c_k\}$ are related to $\{C(s)\}$ by a Fourier transform (see just before Eqn. 7). To compute which choice of $\{C(s)\}$ maximizes $R$ for a given problem, we used a gradient ascent algorithm several times, each time using a different random set of initial $\{C(s)\}$ values. For the problems whose solutions are exhibited in Figs. 2b and 2c, multiple starting points usually yielded the same solution to within the error tolerance specified for the algorithm [apart from an arbitrary factor by which all of the $C(s)$'s can be multiplied without affecting $R$], and that solution had the largest $R$ of any obtained for the given problem. That is, a limitation sometimes associated with gradient ascent algorithms -- namely, that they may yield multiple "solutions" that are local, but far from global, maxima -- did not appear to be a difficulty in these cases.

Fig. 2b (solid curve) shows the infomax solution for an example having $Q^L(s) = \exp[-(s/s_0)^2]$ and $g(s) = \exp[(s/s_1)^2]$ with $s_0 = 4$, $s_1 = 6$, $N = N' = 32$, and $B_0 = 0.1$. There is a central excitatory peak flanked by shallow inhibitory sidelobes (and weaker additional oscillations). (As noted, the negative of this solution, having a central inhibitory region and excitatory sidelobes, gives the same $R$.) As $B_0$ is increased (a range from 0.001 to 20 was studied), the peak broadens, the sidelobes become shallower (relative to the peak), and the receptive fields of nearby M cells increasingly overlap. This behavior is an example of the "redundancy-diversity" tradeoff discussed in [Linsker, 1988].

**Model D -- Bounded output variance**

Our previous models all produce output values $M_n$ whose variance is not explicitly constrained. More biologically realistic cells have limited output variance. For example, a cell's firing rate must lie between zero and some maximum value. Thus, the output of a model nonlinear cell is often taken to be a sigmoid function of $(\Sigma_i C_{ni}L_i)$.

Within the context of linear cell models, we can capture the effect of a bounded output variance by using Model D of Fig. 1. We pass the intermediate output $\Sigma_i C_{ni}(L_i + \nu_{ni})$ through a gain control GC that normalizes the output variance to unity, then we add a final (i.i.d. Gaussian) noise term $\nu'_n$ of variance $B_1$. That is,

$$M_n = GC[\Sigma_i C_{ni}(L_i + \nu_{ni})] + \nu'_n. \tag{13}$$

Without the last term, this model would be identical to Model C, since multiplying both the signal and the $\nu_{ni}$ noise by the same factor GC would not affect $R$. The last term in effect fixes the number of output values that can be discriminated (i.e., not confounded with each other by the noise process $\nu'_n$) to be of order $B_1^{-1/2}$.

The information rate for this model is derived to be (cf. Eqn. 12):

$$R = (1/2)\Sigma_k \ln\{1 + \lambda_k |c_k|^2 / [B_0 \Sigma_s g(s)C(s)^2 + B_1 V(C)]\} \tag{14}$$

where $V(C)$ is the variance of the intermediate output before it is passed through GC:

$$V(C) = (1/N)\Sigma_k \lambda_k |c_k|^2 + B_0 \Sigma_s g(s)C(s)^2. \tag{15}$$

Fig. 2b (dotted curve) shows the infomax solution (numerically obtained as above) for the same $Q^L(s)$ and $g(s)$ functions and parameter values as were used to generate the solid curve (for Model C), but with the new parameter $B_1 = 0.4$. The effect of the new $B_1$ noise process in this case is to deepen the inhibitory sidelobes (relative to the central peak). The more pronounced center-surround character of the resulting M cell dampens the response of the cell to differences (between different input patterns) in the spatially uniform component of the input pattern. This response property allows the $L \rightarrow M$ mapping to be infomax-optimal when the dynamic range of the cells' output response is constrained. (A competing effect can complicate the analysis: If $B_1$ is increased much further, for example to 50 in the case discussed, the sidelobes move to larger $s$ and become shallower. This behavior resembles that discussed at the end of the previous section for the case of increasing $B_0$; in the present case it is the overall noise level that is being increased when $B_1$ increases and $B_0$ is kept constant.)

**Temporally-correlated input patterns**

Let us see how infomax can be used to extract regularities in input time series, as contrasted with the spatially-correlated input patterns discussed above. We consider a single M cell that, at each discrete time denoted by $n$, can process inputs $\{L_i\}$ from earlier times $i \leq n$ (via delay lines, for example). We use the same Model D as before. There are two differences: First, we want $g(s) = \infty$ for all $s > 0$ (input lines from future times are "infinitely noisy"). [A technical point: Our use of periodic boundary conditions, while computationally convenient, means that the input value that will occur $s$ time steps from now is the same value that occurred $(N - s)$ steps ago. We deal with this by choosing $g(s)$ to equal 1 at $s = 0$, to increase as $s \rightarrow -N/2$ (going into the past), and to increase further as $s$ decreases from $+N/2$ to 1, corresponding to increasingly remote past times. The periodicity causes no unphysical effects, provided that we make $g(s)$ increase rapidly enough (or make $N$ large enough) so that $C(s)$ is negligible for time intervals comparable to $N$.] Second, the fact that $C_{ni}$ is a function only of $s_{ni}$ is now a consequence of the constancy of connection weights $C(s)$ of a single M cell with time, rather than merely a convenient Ansatz to facilitate the infomax computation for a set of many M cells (as it was in previous sections).

The infomax solution is shown in Fig. 2c for an example having $Q^L(s) = \exp[-(s/s_0)^2]$; $g(s) = \exp[-t(s)/s_1]$ with $t(s) \equiv s$ for $s \leq 0$ and $t(s) \equiv s - N$ for $s \geq 1$; $s_0 = 4$, $s_1 = 6$, $N = 32$, $B_0 = 0.1$, and $B_1 = 0.4$. The result is that the "temporal receptive field" of the M cell is excitatory for recent times, and

inhibitory for somewhat more remote times (with additional weaker oscillations). The cell's output can be viewed approximately as a linear combination of a smoothed input and a smoothed first time derivative of the input, just as the output of the center-surround cell of Fig. 2b can be viewed as a linear combination of a smoothed input and a smoothed second spatial derivative of the input. As in Fig. 2b, setting $B_1 = 0$ (not shown) lessens the relative inhibitory contribution.

## SUMMARY

To gain insight into the operation of the principle of maximum information preservation, we have applied the principle to the problem of the optimal design of an array of linear filters under various conditions. The filter models that have been used are motivated by certain features that appear to be characteristic of biological networks. These features include the favoring of short connections and the constrained range of output signal values. When nearby input signals (in space or time) are correlated, the infomax-optimal solutions for the cases studied include (1) center-surround cells and (2) cells sensitive to temporal variations in input. The results of the mathematical analysis presented here apply also to arbitrary input covariance functions of the form $Q^L(|i - j|)$. We have also presented more general expressions for the information rate, which can be used even when $Q^L$ is not of this form. The cases discussed illustrate the operation of the infomax principle in some relatively simple but instructive situations. The analysis and results suggest how the principle may be applied to more biologically realistic networks and input ensembles.

### References

T. Berger, *Rate Distortion Theory* (Prentice-Hall, Englewood Cliffs, N.J., 1971), chap. 4.

R. G. Gallager, *Information Theory and Reliable Communication* (John Wiley and Sons, N.Y., 1968), p. 388.

R. Linsker, in: *Neural Information Processing Systems* (Denver, Nov. 1987), ed. D. Z. Anderson (Amer. Inst. of Physics, N.Y.), pp. 485-494.

R. Linsker, *Computer* **21** (3) 105-117 (March 1988).

C. E. Shannon and W. Weaver, *The Mathematical Theory of Communication* (Univ. of Illinois Press, Urbana, 1949).
